# Data Integration for Classification Problems Employing Gaussian Process Priors

**Mark Girolami**
Department of Computing Science
University of Glasgow
Scotland, UK
girolami@dcs.gla.ac.uk

**Mingjun Zhong**
IRISA, Campus de Beaulieu
F-35042 Rennes Cedex
France
zmingjun@irisa.fr

## Abstract

By adopting Gaussian process priors a fully Bayesian solution to the problem of integrating possibly heterogeneous data sets within a classification setting is presented. Approximate inference schemes employing Variational & Expectation Propagation based methods are developed and rigorously assessed. We demonstrate our approach to integrating multiple data sets on a large scale protein fold prediction problem where we infer the optimal combinations of covariance functions and achieve state-of-the-art performance without resorting to any *ad hoc* parameter tuning and classifier combination.

## 1 Introduction

Various emerging quantitative measurement technologies in the life sciences are producing genome, transcriptome and proteome-wide data collections which has motivated the development of data integration methods within an inferential framework. It has been demonstrated that for certain prediction tasks within computational biology synergistic improvements in performance can be obtained via the integration of a number of (possibly heterogeneous) data sources. In [2] six different data representations of proteins were employed for fold recognition of proteins using Support Vector Machines (SVM). It was observed that certain data combinations provided increased accuracy over the use of any single dataset. Likewise in [9] a comprehensive experimental study observed improvements in SVM based gene function prediction when data from both microarray expression and phylogentic profiles were manually combined. More recently protein network inference was shown to be improved when various genomic data sources were integrated [16] and in [1] it was shown that superior prediction accuracy of protein-protein interactions was obtainable when a number of diverse data types were combined in an SVM. Whilst all of these papers exploited the kernel method in providing a means of data fusion within SVM based classifiers it was initially only in [5] that a means of estimating an optimal linear combination of the kernel functions was presented using semi-definite programming. However, the methods developed in [5] are based on binary SVM's, whilst arguably the majority of important classification problems within computational biology are inherently multiclass. It is unclear how this approach could be extended in a straightforward or practical manner to discrimination over multiple-classes. In addition the SVM is non-probabilistic and whilst *post hoc* methods for obtaining predictive probabilities are available [10] these are not without problems such as overfitting. On the other hand Gaussian Process (GP) methods [11], [8] for classification provide a very natural way to both integrate and infer optimal combinations of multiple heterogeneous datasets via composite covariance functions within the Bayesian framework an idea first proposed in [8].

In this paper it is shown that GP's can indeed be successfully employed on general classification problems, without recourse to *ad hoc* binary classification combination schemes, where there are multiple data sources which are also optimally combined employing full Bayesian inference. A

large scale example of protein fold prediction [2] is provided where state-of-the-art predictive performance is achieved in a straightforward manner without resorting to any extensive *ad hoc* engineering of the solution (see [2], [13]). As an additional important by-product of this work inference employing Variational Bayesian (VB) and Expectation Propagation (EP) based approximations for GP classification over multiple classes are studied and assessed in detail. It has been unclear whether EP based approximations would provide similar improvements in performance in the multi-class setting over the Laplace approximation and this work provides experimental evidence that both Variational and EP based approximations perform as well as a Gibbs sampler consistently outperforming the Laplace approximation. In addition we see that there is no statistically significant practical advantage of EP based approximations over VB approximations in this particular setting.

## 2 Integrating Data with Gaussian Process Priors

Let us denote each of $\mathcal{J}$ independent (possibly heterogeneous) feature representations, $\mathcal{F}_j(X)$, of an object $X$ by $\mathbf{x}_j \ \forall \ j = 1 \cdots \mathcal{J}$. For each object there is a corresponding polychotomous response target variable, $t$, so to model this response we assume an additive generalized regression model. Each distinct data representation of $X$, $\mathcal{F}_j(X) = \mathbf{x}_j$, is nonlinearly transformed such that $f_j(\mathbf{x}_j) : \mathcal{F}_j \mapsto \mathbb{R}$ and a linear model is employed in this new space such that the overall nonlinear transformation is $f(X) = \sum_j \beta_j f_j(\mathbf{x}_j)$.

### 2.1 Composite Covariance Functions

Rather than specifying an explicit functional form for each of the functions $f_j(\mathbf{x}_j)$ we assume that each nonlinear function corresponds to a Gaussian process (GP) [11] such that $f_j(\mathbf{x}_j) \sim GP(\boldsymbol{\theta}_j)$ where $GP(\boldsymbol{\theta}_j)$ corresponds to a GP with trend and covariance functions $m_j(\mathbf{x}_j)$ and $C_j(\mathbf{x}_j, \mathbf{x}'_j; \boldsymbol{\theta}_j)$ where $\boldsymbol{\theta}_j$ denotes a set of hyper-parameters associated with the covariance function. Due to the assumed independence of the feature representations the overall nonlinear function will also be a realization of a Gaussian process defined as $f(X) \sim GP(\boldsymbol{\theta}_1 \cdots \boldsymbol{\theta}_{\mathcal{J}}, \beta_1 \cdots \beta_{\mathcal{J}})$ where now the overall trend and covariance functions follow as $\sum_j \beta_j m_j(\mathbf{x}_j)$ and $\sum_j \beta_j^2 C_j(\mathbf{x}_j, \mathbf{x}'_j; \boldsymbol{\theta}_j)$. For $N$ object samples, $X_1 \cdots X_N$, each defined by the $\mathcal{J}$ feature representations, $\mathbf{x}_j^1 \cdots \mathbf{x}_j^N$, denoted by $\mathbf{X}_j$, with associated class specific response $\mathbf{f}_k = [f_k(X_1) \cdots f_k(X_N)]^\mathsf{T}$ the overall GP prior is a multivariate Normal such that

$$\mathbf{f}_k \mid \mathbf{X}_{j=1 \cdots J}, \boldsymbol{\theta}_{1k}, \cdots \boldsymbol{\theta}_{\mathcal{J}k}, \alpha_{1k} \cdots \alpha_{\mathcal{J}k} \sim \mathcal{N}_{\mathbf{f}_k} \left( \mathbf{0}, \sum_j \alpha_{jk} \mathbf{C}_{jk}(\boldsymbol{\theta}_{jk}) \right) \tag{1}$$

The positive random variables $\beta_{jk}^2$ are denoted by $\alpha_{jk}$, zero-trend GP functions have been assumed and each $\mathbf{C}_{jk}(\boldsymbol{\theta}_{jk})$ is an $N \times N$ matrix with elements $C_j(\mathbf{x}_j^m, \mathbf{x}_j^n; \boldsymbol{\theta}_{jk})$. A GP functional prior, over all possible responses (classes), is now available where possibly heterogeneous data sources are integrated via the composite covariance function. It is then, in principle, a straightforward matter to perform Bayesian inference with this model and no further recourse to *ad hoc* binary classifier combination methods or ancillary optimizations to obtain the data combination weights is required.

### 2.2 Bayesian Inference

As we are concerned with classification problems over possibly multiple classes we employ a multinomial probit likelihood rather than a multinomial logit as it provides a means of developing a Gibbs sampler, and subsequent computationally efficient approximations, for the GP random variables. The Gibbs sampler is to be preferred over the Metropolis scheme as no tuning of a proposal distribution is required. As in [3] the auxiliary variables $y_{nk} = f_k(X_n) + \epsilon_{nk}, \ \epsilon_{nk} \sim \mathcal{N}(0, 1)$ are introduced and the $N \times 1$ dimensional vector of target class values associated with each $X_n$ is given as $\mathbf{t}$ where each element $t_n \in \{1, \cdots, K\}$. The $N \times K$ matrix of GP random variables $f_k(X_n)$ is denoted by $\mathbf{F}$. We represent the $N \times 1$ dimensional columns of $\mathbf{F}$ by $\mathbf{F}_{\cdot, k}$ and the corresponding $K \times 1$ dimensional vectors, $\mathbf{F}_{n, \cdot}$, which are formed by the indexed rows of $\mathbf{F}$. The $N \times K$ matrix of auxiliary variables $y_{nk}$ is represented as $\mathbf{Y}$, where the $N \times 1$ dimensional columns are denoted by $\mathbf{Y}_{\cdot, k}$ and the corresponding $K \times 1$ dimensional vectors are obtained from the rows of $\mathbf{Y}$ as $\mathbf{Y}_{n, \cdot}$. The multinomial probit likelihood [3] is adopted which follows as

$$t_n = j \quad \text{if} \quad y_{nj} = \operatorname*{argmax}_{1 \leq k \leq K} \{y_{nk}\} \tag{2}$$

and this has the effect of dividing $\mathbb{R}^K$ into $K$ non-overlapping $K$-dimensional cones $\mathcal{C}_k = \{\mathbf{y} : y_k > y_i, k \neq i\}$ where $\mathbb{R}^K = \cup_k \mathcal{C}_k$ and so each $P(t_n = i|\mathbf{Y}_{n,\cdot})$ can be represented as $\delta(y_{ni} > y_{nk} \ \forall \ k \neq i)$. Class specific independent Gamma priors, with parameters $\boldsymbol{\varphi}_k$, are placed on each $\alpha_{jk}$ and the individual components of $\boldsymbol{\theta}_{jk}$ (denote $\boldsymbol{\Theta}_k = \{\boldsymbol{\theta}_{jk}, \alpha_{jk}\}_{j=1\ldots\mathcal{J}}$), a further Gamma prior is placed on each element of $\boldsymbol{\varphi}_k$ with overall parameters $\mathbf{a}$ and $\mathbf{b}$ so this defines the full model likelihood and associated priors.

## 2.3 MCMC Procedure

Samples from the full posterior $P(\mathbf{Y}, \mathbf{F}, \boldsymbol{\Theta}_{1\ldots K}, \boldsymbol{\varphi}_{1\ldots K}|X_{1\ldots N}, \mathbf{t}, \mathbf{a}, \mathbf{b})$ can be obtained from the following Metropolis-within-Blocked-Gibbs Sampling scheme indexing over all $n = 1 \cdots N$ and $k = 1 \cdots K$.

$$\mathbf{Y}_{n,\cdot}^{(i+1)}|\mathbf{F}_{n,\cdot}^{(i)}, t_n \quad \sim \quad \mathcal{TN}(\mathbf{F}_{n,\cdot}^{(i)}, \mathbf{I}, t_n) \tag{3}$$

$$\mathbf{F}_{\cdot,k}^{(i+1)}|\mathbf{Y}_{\cdot,k}^{(i+1)}, \boldsymbol{\Theta}_k^{(i)}, X_{1,\cdots,N} \quad \sim \quad \mathcal{N}(\boldsymbol{\Sigma}_k^{(i)}\mathbf{Y}_{\cdot,k}^{(i+1)}, \boldsymbol{\Sigma}_k^{(i)}) \tag{4}$$

$$\boldsymbol{\Theta}_1^{(i+1)}|\mathbf{F}_{\cdot,1}^{(i+1)}, \mathbf{Y}_{\cdot,k}^{(i+1)}, \boldsymbol{\varphi}_1^{(i)}, X_{1,\cdots,N} \quad \sim \quad P(\boldsymbol{\Theta}_k^{(i+1)}) \tag{5}$$

$$\boldsymbol{\varphi}_k^{(i+1)}|\boldsymbol{\Theta}_k^{(i+1)}, a_k, b_k \quad \sim \quad P(\boldsymbol{\varphi}_k^{(i+1)}) \tag{6}$$

where $\mathcal{TN}(\mathbf{F}_{n,\cdot}, \mathbf{I}, t_n)$ denotes a conic truncation of a multivariate Gaussian with location parameters $\mathbf{F}_{n,\cdot}$ and dispersion parameters $\mathbf{I}$ and the dimension indicated by the class value of $t_n$ will be the largest. An accept-reject strategy can be employed in sampling from the conic truncated Gaussian however this will very quickly become inefficient for problems with moderately large numbers of classes and as such a further Gibbs sampling scheme may be required. Each $\boldsymbol{\Sigma}_k^{(i)} = \mathbf{C}_k^{(i)}(\mathbf{I}+\mathbf{C}_k^{(i)})^{-1}$ and $\mathbf{C}_k^{(i)} = \sum_{j=1} \alpha_{jk}^{(i)}\mathbf{C}_{jk}(\boldsymbol{\theta}_{jk}^{(i)})$ with the elements of $\mathbf{C}_{jk}(\boldsymbol{\theta}_{jk}^{(i)})$ defined as $C_j(\mathbf{x}_j^m, \mathbf{x}_j^n; \boldsymbol{\theta}_{jk}^{(i)})$. A Metropolis sub-sampler is required to obtain samples for the conditional distribution over the composite covariance function parameters $P(\boldsymbol{\Theta}_k^{(i+1)})$ and finally $P(\boldsymbol{\varphi}_k^{(i+1)})$ is a simple product of Gamma distributions. The predictive likelihood of a test sample $X_*$ is $P(t_* = k|X_*, X_{1\ldots N}, \mathbf{t}, \mathbf{a}, \mathbf{b})$ which can be obtained by integrating over the posterior and predictive prior such that

$$\int P(t_* = k|\mathbf{f}_*)p(\mathbf{f}_*|\boldsymbol{\Omega}, X_*, X_{1\ldots N})p(\boldsymbol{\Omega}|X_{1\ldots N}, \mathbf{t}, \mathbf{a}, \mathbf{b})d\mathbf{f}_*d\boldsymbol{\Omega} \tag{7}$$

where $\boldsymbol{\Omega} = \mathbf{Y}, \boldsymbol{\Theta}_{1\ldots K}$. A Monte-Carlo estimate is obtained by using samples drawn from the full posterior $\frac{1}{S}\sum_{s=1}^S \int P(t_* = k|\mathbf{f}_*)p(\mathbf{f}_*|\boldsymbol{\Omega}^{(s)}, X_*, X_{1\ldots N})d\mathbf{f}_*$ and the integral over the predictive prior requires further conditional samples, $\mathbf{f}_*^{(l|s)}$, to be drawn from each $p(\mathbf{f}_*|\boldsymbol{\Omega}^{(s)}, X_*, X_{1\ldots N})$ finally yielding a Monte Carlo approximation of $P(t_* = k|X_*, X_{1\ldots N}, \mathbf{t}, \mathbf{a}, \mathbf{b})$

$$\frac{1}{LS}\sum_{l=1}^L\sum_{s=1}^S P\left(t_* = k|\mathbf{f}_*^{(l|s)}\right) = \frac{1}{LS}\sum_{l=1}^L\sum_{s=1}^S E_{p(u)}\left\{\prod_{j \neq k}\Phi\left(u + f_{*,k}^{(l|s)} - f_{*,j}^{(l|s)}\right)\right\} \tag{8}$$

MCMC procedures for GP classification have been previously presented in [8] and whilst this provides a practical means to perform Bayesian inference employing GP's the computational cost incurred and difficulties associated with monitoring convergence and running multiple-chains on reasonably sized problems are well documented and have motivated the development of computationally less costly approximations [15]. A recent study has shown that EP is superior to the Laplace approximation for binary classification [4] and that for multi-class classification VB methods are superior to the Laplace approximation [3]. However the comparison between Variational and EP based approximations for the multi-class setting have not been considered in the literature and so we seek to address this issue in the following sections.

## 2.4 Variational Approximation

From the conditional probabilities which appear in the Gibbs sampler it can be seen that a mean field approximation gives a simple iterative scheme which provides a computationally efficient alternative to the full sampler (including the Metropolis sub-sampler for the covariance function parameters),

details of which are given in [3]. However given the excellent performance of EP on a number of approximate Bayesian inference problems it is incumbent on us to consider an EP solution here. We should point out that only the top level inference on the GP variables is considered here and the composite covariance function parameters will be obtained using another appropriate type-II maximum likelihood optimization scheme if possible.

## 2.5 Expectation Propagation with Full Posterior Covariance

The required posterior can also be approximated by EP [7]. In this case the multinomial probit likelihood is approximated by a multivariate Gaussian such that $p(\mathbf{F}|\mathbf{t}, X_{1 \cdots N}) \approx Q(\mathbf{F}) = \prod_k p(\mathbf{F}_{\cdot,k}|X_{1 \cdots N}) \prod_n g_n(\mathbf{F}_{n,\cdot})^1$ where $g_n(\mathbf{F}_{n,\cdot}) = \mathcal{N}_{\mathbf{F}_{n,\cdot}}(\boldsymbol{\mu}_n, \boldsymbol{\Lambda}_n)$, $\boldsymbol{\mu}_n$ is a $K \times 1$ vector and $\boldsymbol{\Lambda}_n$ is a full $K \times K$ dimensional covariance matrix. Denoting the cavity density as $Q^{\backslash n}(\mathbf{F}) = \prod_k p(\mathbf{F}_{\cdot,k}|X_{1 \cdots N}) \prod_{i,i \neq n} g_i(\mathbf{F}_{i,\cdot})$, EP proceeds by iteratively re-estimating the moments $\boldsymbol{\mu}_n, \boldsymbol{\Lambda}_n$ by moment matching [7] giving the following

$$\boldsymbol{\mu}_n^{new} = E_{\hat{p}_n}\{\mathbf{F}_{n,\cdot}\} \ \text{ and } \ \boldsymbol{\Lambda}_n^{new} = E_{\hat{p}_n}\{\mathbf{F}_{n,\cdot}\mathbf{F}_{n,\cdot}^\mathsf{T}\} - E_{\hat{p}_n}\{\mathbf{F}_{n,\cdot}\}E_{\hat{p}_n}\{\mathbf{F}_{n,\cdot}\}^\mathsf{T}, \tag{9}$$

where $\hat{p}_n = \mathcal{Z}_n^{-1}Q^{\backslash n}(\mathbf{F}_{n,\cdot})p(t_n|\mathbf{F}_{n,\cdot})$, and $\mathcal{Z}_n$ is the required normalizing (partition) function which is required to obtain the above mean and covariance estimates. To proceed an analytic form for the partition function $\mathcal{Z}_n$ is required. Indeed for binary classification employing a binomial probit likelihood an elegant EP solution follows due to the analytic form of the partition function [4]. However for the case of multiple classes with a multinomial probit likelihood the partition function no longer has a closed analytic form and further approximations are required to make any progress. There are two strategies which we consider, the first retains the full posterior coupling in the covariance matrices $\boldsymbol{\Lambda}_n$ by employing Laplace Propagation (LP) [14] and the second assumes no posterior coupling in $\boldsymbol{\Lambda}_n$ by setting this as a diagonal covariance matrix. The second form of approximation has been adopted in [12] when developing a multi-class version of the Informative Vector Machine (IVM) [6]. In the first case where we employ LP an additional significant $\mathcal{O}(K^3 N^3)$ computational scaling will be incurred however it can be argued that the retention of the posterior coupling is important. For the second case clearly we lose this explicit posterior coupling but, of course, do not incur the expensive computational overhead required of LP. We observed in unreported experiments that there is little of statistical significance lost, in terms of predictive performance, when assuming a factorable form for each $\hat{p}_n$. LP proceeds by propagating the approximate moments such that

$$\boldsymbol{\mu}_n^{new} \approx \underset{\mathbf{F}_{n,\cdot}}{\mathrm{argmax}} \ \log \hat{p}_n \ \text{ and } \ \boldsymbol{\Lambda}_n^{new} \approx \left[-\frac{\partial^2 \log \hat{p}_n}{\partial \mathbf{F}_{n,\cdot}\partial \mathbf{F}_{n,\cdot}^\mathsf{T}}\right]^{-1} \tag{10}$$

The required derivatives follow straightforwardly and details are included in the accompanying material. The approximate predictive distribution for a new data point $\mathbf{x}_*$ requires a Monte Carlo estimate employing samples drawn from a $K$-dimensional multivariate Gaussian for which details are given in the supplementary material[2].

## 2.6 Expectation Propagation with Diagonal Posterior Covariance

By assuming a factorable approximate posterior, as in the variational approximation [3], a distinct simplification of the problem setting follows, where now we assume that $g_n(\mathbf{F}_{n,\cdot}) = \prod_k \mathcal{N}_{\mathbf{F}_{n,k}}(\mu_{n,k}, \lambda_{n,k})$ i.e. is a factorable distribution. This assumption has already been made in [12] in developing an EP based multi-class IVM. Now significant computational simplification follows where the required moment matching amounts to $\mu_{nk}^{new} = E_{\hat{p}_{nk}}\{\mathbf{F}_{n,k}\}$ and $\lambda_{nk}^{new} = E_{\hat{p}_{nk}}\{\mathbf{F}_{n,k}^2\} - E_{\hat{p}_{nk}}\{\mathbf{F}_{n,k}\}^2$ where the density $\hat{p}_{nk}$ has a partition function which now has the analytic form

$$\mathcal{Z}_n = E_{p(u)p(v)}\left\{\prod_{j=1,j \neq i}^{K} \Phi\left(\frac{u + v\sqrt{\lambda_{ni}^{\backslash n}} + \mu_{ni}^{\backslash n} - \mu_{nj}^{\backslash n}}{\sqrt{1 + \lambda_{nj}^{\backslash n}}}\right)\right\} \tag{11}$$

where $u$ and $v$ are both standard Normal random variables ($v\sqrt{\lambda_{ni}^{\backslash n}} = \mathbf{F}_{n,i} - \mu_{ni}^{\backslash n}$) with $\lambda_{ni}^{\backslash n}$ and $\mu_{ni}^{\backslash n}$ having the usual meanings (details in accompanying material). Derivatives of this partition function follow in a straightforward way now allowing the required EP updates to proceed (details in supplementary material). The approximate predictive distribution for a new data point $X_*$ in this case takes a similar form to that for the Variational approximation [3]. So we have

$$P(t_* = k | X_*, X_{1\cdots N}, \mathbf{t}) = E_{p(u)p(v)} \left\{ \prod_{j=1, j \neq k}^{K} \Phi\left( \frac{u + v\sqrt{\lambda_k^*} + \mu_k^* - \mu_j^*}{\sqrt{1 + \lambda_j^*}} \right) \right\} \qquad (12)$$

where the predictive mean and variance follow in standard form.

$$\mu_j^* = (\mathbf{C}_j^*)^{\mathsf{T}} (\mathbf{C}_j + \mathbf{\Lambda}_j)^{-1} \boldsymbol{\mu}_j \ \text{ and } \ \lambda_j^* = c_j^* - (\mathbf{C}_j^*)^{\mathsf{T}} (\mathbf{C}_j + \mathbf{\Lambda}_j)^{-1} \mathbf{C}_j^* \qquad (13)$$

It should be noted here that the expectation over $p(u)$ and $p(v)$ could be computed by using either Gaussian quadrature or a simple Monte Carlo approximation which is straightforward as sampling from a univariate standardized Normal only is required. The VB approximation [3] however only requires a 1-D Monte Carlo integral rather than the 2-D one required here.

## 3  Experiments

Before considering the main example of data integration within a large scale protein fold prediction problem we attempt to assess a number of approximate inference schemes for GP multi-class classification. We provide a short comparative study of the Laplace, VB, and both possible EP approximations by employing the Gibbs sampler as the comparative gold standard. For these experiments six multi-class data sets are employed [3], i.e., Iris ($N = 150$, $K = 3$), Wine ($N = 178$, $K = 3$), Soybean ($N = 47$, $K = 4$), Teaching ($N = 151$, $K = 3$), Waveform ($N = 300$, $K = 3$) and ABE ($N = 300$, $K = 3$, which is a subset of the *Isolet* dataset using the letters 'A', 'B' and 'E',). A single radial basis covariance function with one length scale parameter is used in this comparative study. Ten-fold cross validation (CV) was used to estimate the predictive log-likelihood and the percentage predictive error. Within each of the ten folds a further 10 CV routine was employed to select the length-scale of the covariance function. For the Gibbs sampler, after a burn-in of 2000 samples, the following 3000 samples were used for inference, and the predictive error and likelihood were computed from the 3000 post-burn-in samples. For each data set and each method the percentage predictive error and the predictive log-likelihood were estimated in this manner. The summary results given as the mean and standard deviation over the ten folds are shown in Table 1. The results which cannot be distinguished from each other, under a Wilcoxon rank sum test with a 5% significance level, are highlighted in bold. From those results, we can see that across most data sets used, the predictive log-likelihood obtained from the Laplace approximation is lower than those of the three other methods. In our observations, the predictive performance of VB and the IEP approximation are consistently indistinguishable from the performance achieved from the Gibbs sampler. From the experiments conducted there is no evidence to suggest any difference in predictive performance between IEP & VB methods in the case of multi-way classification. As there is no benefit in choosing an EP based approximation over the Variational one we now select the Variational approximation in that inference over the covariance parameters follows simply by obtaining posterior mean estimates using an importance sampler.

As a brief illustration of how the Variational approximation compares to the full Metropolis-within-Blocked-Gibbs Sampler consider a toy dataset consisting of three classes formed by a Gaussian surrounded by two annular rings having ten features only two of which are predictive of the class labels [3]. We can compare the compute time taken to obtain reasonable predictions from the full MCMC and the approximate Variational scheme [3]. Figure 1 (a) shows the samples of the covariance function parameters $\mathbf{\Theta}$ drawn from the Metropolis subsampler[4] and overlaid in black the corresponding approximate posterior mean estimates obtained from the variational scheme [3]. It

Table 1: Percentage predictive error (PE) and predictive log-likelihood (PL) for six data sets from UCI computed using Laplace, Variational Bayes (VB), independent EP (IEP), as well as MCMC using Gibbs sampler. Best results which are statistically indistinguishable from each other are highlighted in bold.

| | ABE | | Iris | |
|---|---|---|---|---|
| | PE | PL | PE | PL |
| Laplace | 4.000±3.063 | -0.290±0.123 | 3.333±3.513 | -0.132±0.052 |
| VB | 2.000±2.330 | **-0.164±0.026** | 3.333±3.513 | **-0.087±0.056** |
| Gibbs | 3.333±3.143 | **-0.158±0.037** | 3.333±3.513 | **-0.079±0.056** |
| IEP | 5.333±5.019 | **-0.139±0.050** | 3.333±3.513 | **-0.063±0.059** |
| | Wine | | Soybean | |
| | PE | PL | PE | PL |
| Laplace | 3.889±5.885 | -0.258±0.045 | 0.000±0.000 | -0.359±0.040 |
| VB | 2.222±3.884 | **-0.182±0.057** | 0.000±0.000 | **-0.158±0.034** |
| Gibbs | 4.514±5.757 | **-0.177±0.054** | 0.000±0.000 | **-0.158±0.039** |
| IEP | 3.889±5.885 | **-0.133±0.047** | 0.000±0.000 | **-0.172±0.037** |
| | Teach | | Wave | |
| | PE | PL | PE | PL |
| Laplace | 39.24±15.74 | -0.836±0.072 | 17.50±9.17 | **-0.430±0.085** |
| VB | 41.12±9.92 | **-0.711±0.125** | 18.33±9.46 | **-0.410±0.100** |
| Gibbs | 42.41±6.22 | **-0.730±0.113** | 15.83±8.29 | **-0.380±0.116** |
| IEP | 42.54±11.32 | **-0.800±0.072** | 17.50±10.72 | **-0.383±0.107** |

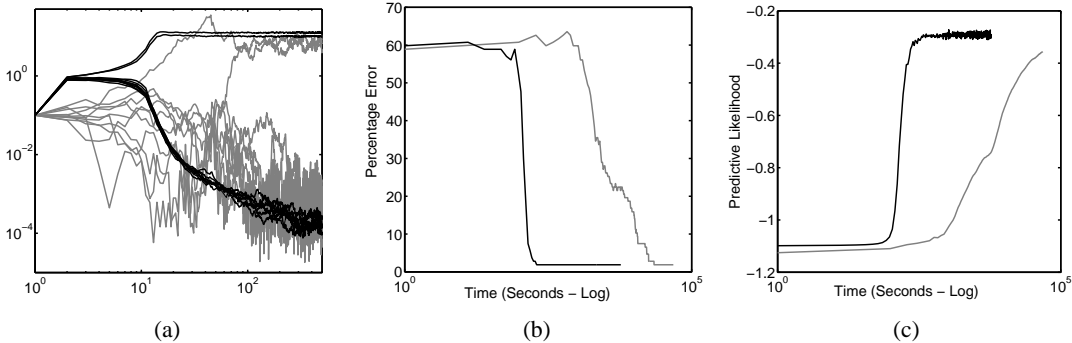

Figure 1: (a) Progression of MCMC and Variational methods in estimating covariance function parameters, vertical axis denotes each $\theta_d$, horizontal axis is time (all log scale) (b) percentage error under the MCMC (gray) and Variational (black) schemes, (c) predictive likelihood under both schemes.

is clear that after 100 calls to the sub-sampler the samples obtained reflect the relevance of the features, however the deterministic steps taken in the variational routine achieve this in just over ten computational steps of equal cost to the Metropolis sub-sampler. Figure 1 (b) shows the predictive error incurred by the classifier and under the MCMC scheme 30,000 CPU seconds are required to achieve the same level of predictive accuracy under the variational approximation obtained in 200 seconds (a factor of 150 times faster). This is due, in part, to the additional level of sampling from the predictive prior which is required when using MCMC to obtain predictive posteriors. Because of these results we now adopt the variational approximation for the following large scale experiment.

## 4   Protein Fold Prediction with GP Based Data Fusion

To illustrate the proposed GP based method of data integration a substantial protein fold classification problem originally studied in [2] and more recently in [13] is considered. The task is to devise a predictor of 27 distinct SCOP classes from a set ($N = 314$) of low homology protein sequences. Six

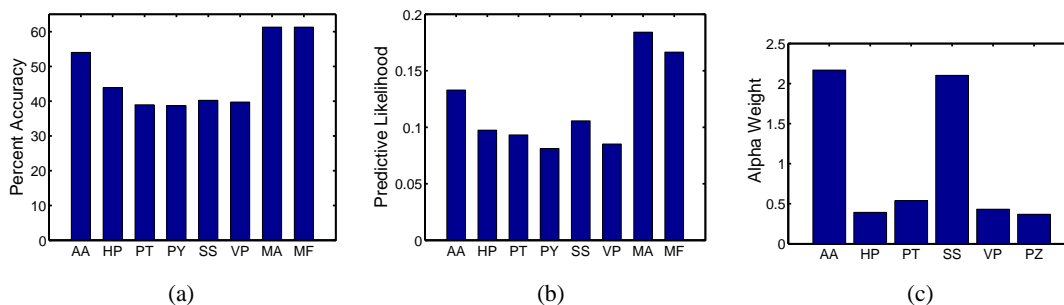

Figure 2: (a) The prediction accuracy for each individual data set and the corresponding combinations, (MA) employing inferred weights and (MF) employing a fixed weighting scheme (b) The predictive likelihood achieved for each individual data set and with the integrated data (c) The posterior mean values of the covariance function weights $\alpha_1 \cdots \alpha_6$.

different data representations (each comprised of around 20 features) are available characterizing (1) Amino Acid composition (AA); (2) Hydrophobicity profile (HP); (3) Polarity (PT); (4) Polarizability (PY); (5) Secondary Structure (SS); (6) Van der Waals volume profile of the protein (VP). In [2] a number of classifier and data combination strategies were employed in devising a multiway classifier from a series of binary SVM's. In the original work of [2] the best predictive accuracy obtained on an independent set ($N = 385$) of low sequence similarity proteins was 53%. It was noted after extensive careful manual experimentation by the authors that a combination of Gaussian kernels each composed of the (AA), (SS) and (HP) datasets significantly improved predictive accuracy. More recently in [13] a heavily tuned *ad hoc* ensemble combination of classifiers raised this performance to 62% the best reported on this problem. We employ the proposed GP based method (Variational approximation) in devising a classifier for this task where now we employ a composite covariance function (shared across all 27 classes), a linear combination of RBF functions for each data set. Figure (2) shows the predictive performance of the GP classifier in terms of percentage prediction accuracy (a) and predictive likelihood on the independent test set (b). We note a significant synergistic increase in performance when all data sets are combined and weighted (MA) where the overall performance accuracy achieved is 62%. Although the 0-1 loss test error is the same for an equal weighting of the data sets (MF) and that obtained using the proposed inference procedure (MA) for (MA) there is an increase in predictive likelihood i.e. more confident correct predictions being made. It is interesting to note that the weighting obtained (posterior mean for $\alpha$) Figure (2.c) weights the (AA) & (SS) with equal importance whilst other data sets play less of a role in performance improvement.

## 5 Conclusions

In this paper we have considered the problem of integrating data sets within a classification setting, a common scenario within many bioinformatics problems. We have argued that the GP prior provides an elegant solution to this problem within the Bayesian inference framework. To obtain a computationally practical solution three approximate approaches to multi-class classification with GP priors, i.e. Laplace, Variational and EP based approximations have been considered. It is found that EP and Variational approximations approach the performance of a Gibbs sampler and indeed their predictive performances are indistinguishable at the 5% level of significance. The full EP (FEP) approximation employing LP has an excessive computational cost and there is little to recommend it in terms of predictive performance over the independent assumption (IEP). Likewise there is little to distinguish between IEP and VB approximations in terms of predictive performance in the multi-class classification setting though further experiments on a larger number of data sets is desirable. We employ VB to infer the optimal parameterized combinations of covariance functions for the protein fold prediction problem over 27 possible folds and achieve state-of-the-art performance without recourse to any *ad hoc* tinkering and tuning and the inferred combination weights are intuitive in terms of the information content of the highest weighted data sets. This is a highly practical solution to the problem of heterogenous data fusion in the classification setting which employs Bayesian inferen-

tial semantics throughout in a consistent manner. We note that on the fold prediction problem the best performance achieved is equaled without resorting to complex and *ad hoc* data and classifier weighting and combination schemes.

## 5.1 Acknowledgements

MG is supported by the Engineering and Physical Sciences Research Council (UK) grant number EP/C010620/1, MZ is supported by the National Natural Science Foundation of China grant number 60501021.

## Footnotes

[1]Conditioning on the covariance function parameters and associated hyper-parameters is implicit

[2]Supplementary material `http://www.dcs.gla.ac.uk/people/personal/girolami/pubs_2006/NIPS2006/index.htm`

[3] http://www.ics.uci.edu/~mlearn/MPRepository.html

[4] It should be noted that multiple Metropolis sub-chains had to be run in order to obtain reasonable sampling of the $\mathbf{\Theta} \in \mathbb{R}_+^{10}$

## References

[1] A. Ben-Hur and W.S. Noble. Kernel methods for predicting protein-protein interactions. *Bioinformatics*, 21, Suppl. 1:38–46, 2005.

[2] Chris Ding and Inna Dubchak. Multi-class protein fold recognition using support vector machines and neural networks. *Bioinformatics*, 17:349–358, 2001.

[3] Mark Girolami and Simon Rogers. Variational Bayesian multinomial probit regression with Gaussian process priors. *Neural Computation*, 18(8):1790–1817, 2006.

[4] M. Kuss and C.E. Rasmussen. Assessing approximate inference for binary Gaussian process classification. *Journal of Machine Learning Research*, 6:1679–1704, 2005.

[5] G. R. G. Lanckriet, T. De Bie, N. Cristianini, M. I. Jordan, and W. S. Noble. A statistical framework for genomic data fusion. *Bioinformatics*, 20:2626–2635, 2004.

[6] Neil Lawrence, Matthias Seeger, and Ralf Herbrich. Fast sparse Gaussian process methods: The informative vector machine. In S. Thrun S. Becker and K. Obermayer, editors, *Advances in Neural Information Processing Systems 15*. MIT Press.

[7] Thomas Minka. *A family of algorithms for approximate Bayesian inference*. PhD thesis, MIT, 2001.

[8] R. Neal. Regression and classification using Gaussian process priors. In A.P. Dawid, M. Bernardo, J.O. Berger, and A.F.M. Smith, editors, *Bayesian Statistics 6*, pages 475–501. Oxford University Press, 1998.

[9] Paul Pavlidis, Jason Weston, Jinsong Cai, and William Stafford Noble. Learning gene functional classifications from multiple data types. *Journal of Computational Biology*, 9(2):401–411, 2002.

[10] J.C. Platt. Probabilities for support vector machines. In A. Smola, P. Bartlett, B. Schlkopf, and D. Schuurmans, editors, *Advances in Large Margin Classifiers*, pages 61–74. MIT Press, 1999.

[11] Carl Edward Rasmussen and Christopher K. I. Williams. *Gaussian Processes for Machine Learning*. MIT Press, 2006.

[12] M.W. Seeger, N.D. Lawrence, and R. Herbrich. Efficient nonparametric Bayesian modelling with sparse Gaussian process approximations. *Technical Report*, "http://www.kyb.tuebingen.mpg.de/bs/people/seeger/", 2006.

[13] Hong-Bin Shen and Kuo-Chen Chou. Ensemble classifier for protein fold pattern recognition. *Bioinformatics*, Advanced Access(doi:10.1093), 2006.

[14] Alexander Smola, Vishy Vishwanathan, and Eleazar Eskin. Laplace propagation. In Sebastian Thrun, Lawrence Saul, and Bernhard Schölkopf, editors, *Advances in Neural Information Processing Systems 16*. MIT Press, Cambridge, MA, 2004.

[15] C.K.I. Williams and D. Barber. Bayesian classification with Gaussian processes. *IEEE Transactions on Pattern Analysis and Machine Intelligence*, 20(12):1342–1352, 1998.

[16] Y. Yamanishi, J. P. Vert, and M. Kanehisa. Protein network inference from multiple genomic data: a supervised approach. *Bioinformatics*, 20, Suppl. 1:363–370, 2004.
